# Homeostasis in a Silicon Integrate and Fire Neuron

**Shih-Chii Liu**
Institute for Neuroinformatics, ETH/UNIZ
Winterthurstrasse 190, CH-8057 Zurich
Switzerland
shih@ini.phys.ethz.ch

**Bradley A. Minch**
School of Electrical and Computer Engineering
Cornell University
Ithaca, NY 14853-5401, U.S.A.
minch@ee.cornell.edu

## Abstract

In this work, we explore homeostasis in a silicon integrate-and-fire neuron. The neuron adapts its firing rate over long time periods on the order of seconds or minutes so that it returns to its spontaneous firing rate after a lasting perturbation. Homeostasis is implemented via two schemes. One scheme looks at the presynaptic activity and adapts the synaptic weight depending on the presynaptic spiking rate. The second scheme adapts the synaptic "threshold" depending on the neuron's activity. The threshold is lowered if the neuron's activity decreases over a long time and is increased for prolonged increase in postsynaptic activity. Both these mechanisms for adaptation use floating-gate technology. The results shown here are measured from a chip fabricated in a 2-$\mu$m CMOS process.

## 1 Introduction

We explored long-time constant adaptation mechanisms in a simple integrate-and-fire silicon neuron. Many researchers have postulated constant adaptation mechanisms which, for example, preserve the firing rate of the neuron over long time invervals (Liu et al. 1998) or use the presynaptic spiking statistics to adapt the spiking rate of the neuron so that the distribution of this spiking rate is uniformly distributed (Stemmler and Koch 1999). Homeostasis is observed in in-vitro recordings (Desai et al. 1999) where if the K or Na conductances are perturbed by adding antagonists, the cell returns to its original spiking rate in a couple of days.

This work differs from previous work that explore the adaptation of the firing threshold and the gain of the neuron through the regulation of Hodgkin-Huxley like conductances (Shin and Koch 1999) and regulation of the neuron to perturbation in the conductances (Simoni and DeWeerth 1999). Our neuron circuit is a simple integrate-and-fire neuron and

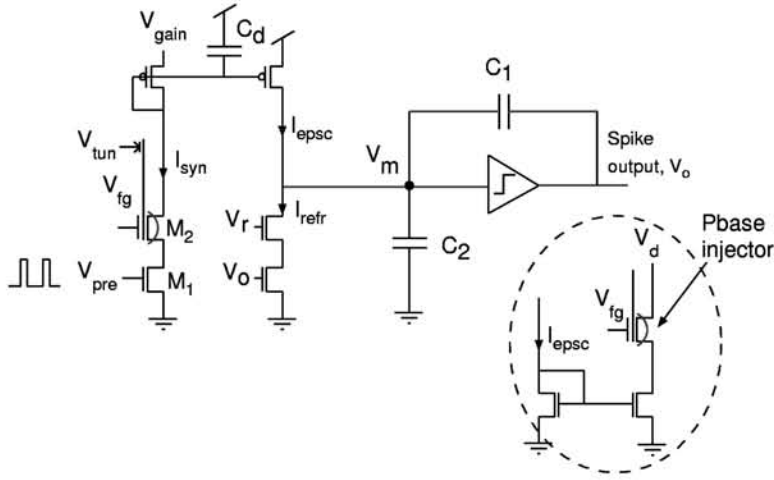

Figure 1: Schematic of neuron circuit with long time constant mechanisms for presynaptic adaptation.

our adaptation mechanisms have time constants of seconds to minutes. We also describe adaptation of the synaptic weight to presynaptic spiking rates. This presynaptic adaptation models the contrast gain control curves of cortical simple cells (Ohzawa et al. 1985).

We fabricated two different circuits in a 2-$\mu$m CMOS process. One circuit implements presynaptic adaptation and the other circuit implements postsynaptic adaptation. The long time constant adaptation mechanisms use tunnelling and injection mechanisms to remove charge from and to add charge onto a floating gate (Diorio et al. 1999). We added these mechanisms to a simple integrate-and-fire neuron circuit (Mead 1989). This circuit (shown in Figure 1) takes an input current, $I_{epsc}$, which charges up the membrane, $V_m$. When the membrane exceeds a threshold, the output of the neuron, $V_o$, spikes. The spiking rate of the neuron, $f_o$ is determined by the input current, $I_{epsc}$, that is, $f_o = m\, I_{epsc}$ where $m = \frac{1}{(C_1 + C_2)V dd}$ is a constant.

## 2 Adaptation mechanisms in silicon neuron circuit

In order to permit continuous operation with only positive polarity bias voltages, we use two distinct mechanisms to modify the floating-gate charges in our neuron circuits. We use Fowler-Nordheim tunneling through high-quality gate oxide to remove electrons from the floating gates (Lenzlinger and Snow 1969). Here, we apply a large voltage across the oxide, which reduces the width of the Si-SiO$_2$ energy barrier to such an extent that electrons are likely to tunnel through the barrier. The tunneling current is given approximately by

$$I_{tun} = I_{0t} e^{-V_o/V_{ox}},$$

where $V_{ox} = V_{tun} - V_{fg}$ is the voltage across the tunneling oxide and $I_{0t}$ and $V_o$ are measurable device parameters. For the 400-Å oxides that are typical of a 2-$\mu$m CMOS process, a typical value of $V_o$ is 1000 V and an oxide voltage of about 30 V is required to obtain an appreciable tunneling current.

We use subthreshold channel hot-electron injection in an $n$MOS transistor (Diorio, Minch, and Hasler 1999) to add electrons to the floating gates. In this process, electrons in the channel of the $n$MOS transistor accelerate in the high electric field that exists in the depletion region near the drain, gaining enough energy to surmount the Si-SiO$_2$ energy barrier

(about 3.2 eV). To facilitate the hot-electron injection process, we locally increase the substrate doping density of the $n$MOS transistor using the $p$-base layer that is normally used to form the base of a vertical $npn$ bipolar transistor. The $p$-base substrate implant simultaneously increases the electric field at the drain end of the channel and increases the $n$MOS transistor's threshold voltage from 0.8 V to about 6 V, permitting subthreshold operation at gate voltages that permit the collection of the injected electrons by the floating gate. The hot-electron injection current is given approximately by

$$I_{inj} = \eta I_s e^{\phi_{dc}/V_{inj}},$$

where $I_s$ is the source current, $\phi_{dc}$ is the drain-to-channel voltage, and $\eta$ and $V_{inj}$ are measurable device parameters. The value of $V_{inj}$ is a bias dependent injection parameter and typically ranges from 60 mV to 0.1 V.

## 3 Presynaptic adaptation

The first mechanism adapts the synaptic efficacy to the presynaptic firing rate over long time constants. The circuit for this adaptation mechanism is shown in Figure 1. The synaptic current is generated by a series of two transistors; one is driven by the presynaptic input and the other by the floating-gate voltage. The floating-gate voltage stores the synaptic efficacy of the synapse. A discrete amount of charge is integrated on a diode capacitor every time there is a presynaptic spike. The charge that is dumped onto the capacitor depends on the input frequency and the synaptic weight. The excitatory postsynaptic current to the membrane of the neuron depends also on the gain of the current-mirror. The tunneling mechanism which is controlled by $V_{tun}$ is continuously on so the synaptic efficacy slowly decreases over time. The injection mechanism is turned on only when there is a presynaptic spike. This presynaptic adaptation can model the contrast gain control curves of cortical simple cells.

### 3.1 Steady-state analysis

In steady-state, the tunneling current, $I_{tun}$, is equal to the average injection current, $I_{inj}$ and they are as follows:

$$I_{tun} = I_{ot} e^{-\frac{V_o}{V_{tun}-V_{fg0}}} \tag{1}$$

$$I_{inj} = (e^{\frac{I_{opb}e^{kV_{fg0}/U_T}T_\delta}{Q_T}} - 1)AQ_T f_i \tag{2}$$

where $A$ is the gain of the current mirror integrator, $Q_T = C_d U_T/k$, $V_{fg0}$ is the steady-state floating-gate voltage, $f_i$ is the presynaptic rate and $T_\delta$ is the pulse width of the presynaptic pulse. From Equations 1 and 2, we can solve for $V_{fg0}$ and thus determine the synaptic current, $I_{syn}$:

$$I_{syn} = I_{opb} e^{\frac{kV_{fg0}}{U_T}} = I_m/(f_i T_\delta)^{\frac{1}{\beta}}.$$

In this equation, $I_m$ is a preconstant and $\beta$ is approximately 1. The steady-state input current is given by $I_{epsc} = I_{syn} T_\delta A f_i \approx I_m A$, thus it is independent of the presynaptic input frequency.

### 3.2 Transient analysis

With a transient change in the presynaptic frequency, $f_i$, the initial postsynaptic frequency is given by:

$$f_o + df_o = m * \frac{I_m A(f_i + df_i)}{f_i} = f_o + m * I_m A(df_i/f_i). \tag{3}$$

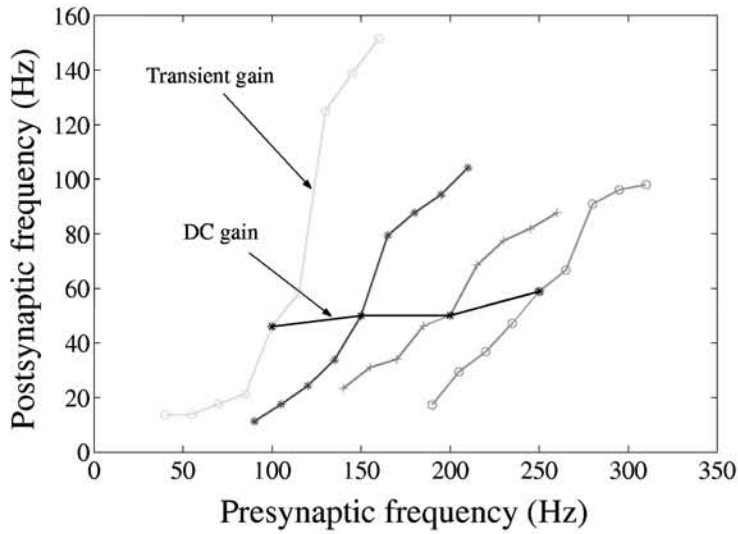

Figure 2: Adaptation curves of synaptic efficacy to presynaptic frequencies using long time constant adaptation mechanisms.

As derived from Equation 3, we see that the transient change in the neuron's spiking rate is dependent on the contrast of the input spiking rate, $df_i/f_i$.

$$df_o = m * I_m * A * df_i/f_i = f_o(df_i/f_i)$$
$$\Rightarrow df_o/df_i = f_o/f_i \tag{4}$$

Hence, the transient gain of the neuron is equal to the ratio of the postsynaptic spiking rate to the presynaptic input rate and it decreases with the input rate.

### 3.3  Experimental results

We measured the transient and steady-state spiking rates of the neuron around four different steady-state presynaptic rates of 100Hz, 150Hz, 200Hz, and 250Hz. In these measurements, the drain of the pbase injection transistor was set at 4V and the tunnelling voltage was set at 35.3V. For each steady-state presynaptic rate, we presented step increases and decreases in the presynaptic rate of 15Hz, 30Hz, 45Hz, and 60Hz. The instantaneous postsynaptic rate is plotted along one the four steep curves in Figure 2. After every change in the presynaptic rate, we returned the presynaptic rate to its steady-state value before we presented the next change in presynaptic rate. The transient gain of the curves decreases for higher input spiking rates. This is predicted by Equation 4.

We also recorded the dynamics of the adaptation mechanisms by measuring the spiking rate of the neuron when the presynaptic frequency was decreased at time (t=0) from 350 Hz to 300 Hz as shown in Figure 3. The system adapts over a time constant of minutes back to the initial output frequency. These data show that the synaptic efficacy adapted to a higher weight value over time. The time constant of adaptation can be increased by either increasing the tunnelling voltage or the pbase injector's drain voltage, $V_d$.

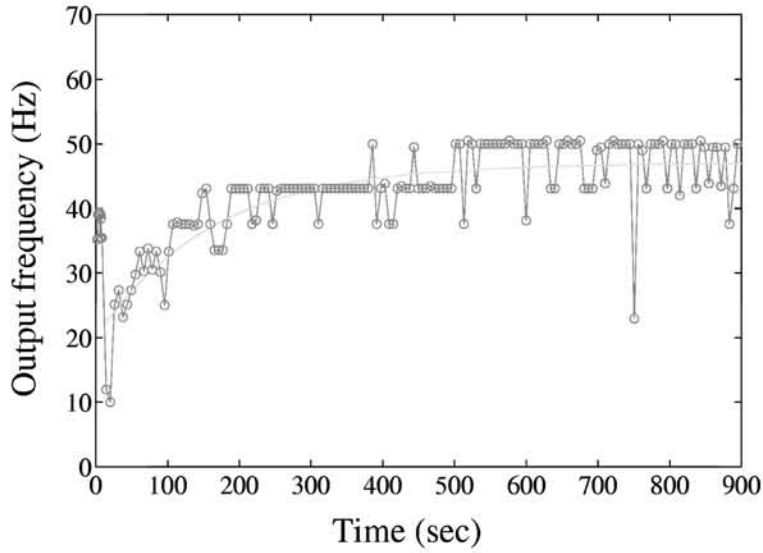

Figure 3: Temporal adaptation of spiking rate of neuron to a decrease in the presynaptic frequency from 350Hz to 300Hz. The smooth line is an exponential fit to the data curve.

## 4    Postsynaptic adaptation

In the second mechanism, the neuron's spiking rate determines the synaptic "threshold". The schematic of this adaptation circuitry is shown in Figure 4. The floating-gate pbase transistor provides a quiescent input to the neuron so that the neuron fires at a quiescent rate. The tunneling mechanism is always turned on so the neuron's spiking rate increases in time if the neuron does not spike. However the injection mechanism turns on when the neuron spikes. The time constant of these mechanisms is in terms of seconds to minutes. The increase in the floating-gate voltage is equivalent to a decrease in the synaptic threshold. If the neuron's activity is high, the injection mechanism turns on thus decreasing the floating-gate voltage and the input current to the neuron. These two opposing mechanisms ensure that the cell will remain at a constant activity under steady-state conditions. In other words, the threshold of the neuron is modulated by its output spiking rate. The threshold of the neuron continuously decreases and each output spike increases the threshold.

### 4.1    Steady-state analysis

Similar equations as in Section 3.1 can be used to solve for $V_{fg0}$, thus leading us to the following expression for the steady-state input current, $I_{in0}$:

$$I_{in0} = I_{opb}e^{\frac{kV_{fg0}}{U_T}} = I_m/(f_oT_\delta)^\gamma$$

where $I_m$ is a preconstant and $\gamma$ is close to 1.

### 4.2    Transient analysis

When a positive step voltage is applied to $V_{ex}$, the step change, $\Delta V$, is coupled into the floating gate. The initial transient current is:

$$I_{in}(t = 0+) = I_{in0}e^{\frac{k\Delta V}{U_T}}$$

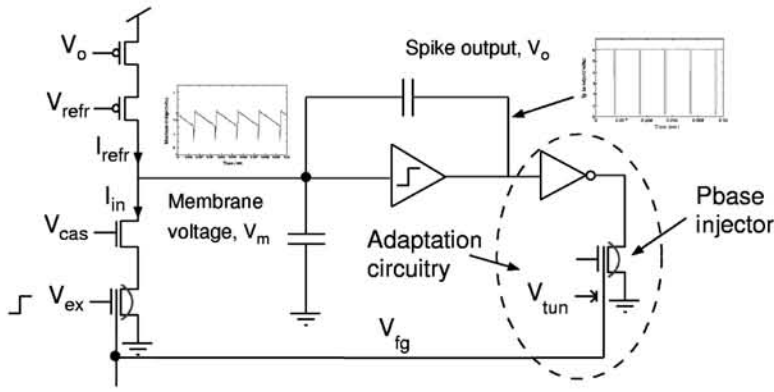

Figure 4: Schematic of neuron circuit with long time constant mechanisms for postsynaptic adaptation.

and the initial increase in the postsynaptic firing rate is

$$f_o + df_o = f_o e^{\frac{k\Delta V}{U_T}}.$$

If we assume that the step input, $V_{in} = \log(f_i)$ (where $f_i$ is the firing rate of the presynaptic neuron), then the change in the floating-gate voltage is described by $\Delta V = df_i/f_i$. We then solve for $df_o$,

$$\frac{df_o}{f_o} = e^{\frac{k\Delta V}{U_T}} - 1 \approx \frac{k}{U_T}\frac{df_i}{f_i}. \tag{5}$$

Equation 5 shows that the transient change in the neuron's spiking rate is proportional to the input contrast in the firing rate. With time, the floating-gate voltage adapts back to the steady-state condition, so the spiking rate returns to $f_o$.

### 4.3 Experimental results

In these experiments, we set the tunneling voltage, $V_{tun}$ to 28V, and the injection voltage to 6.6V. We coupled a step decrease of 0.2V into the floating-gate voltage and then measured the output frequency of the neuron over a period of 10 minutes. The output of this experiment is shown in Figure 5. The frequency dropped from about 19Hz to 13Hz but the circuit adapted after this initial perturbation and the spiking rate of the neuron returned to about 19Hz over 26min. A similar experiment is performed but this time a step increase of 0.2V was coupled into the floating gate node (shown in Figure 5). Initially, the neuron's rate increased from 20Hz to 28Hz but over a long period of minutes, the firing rate returned to 20Hz.

## 5 Conclusion

In this work, we show how long-time constant adaptation mechanisms can be added to a silicon integrate-and-fire neuron in a normal CMOS process. These homeostatic mechanisms can be combined with short time constant synaptic depressing synapses on the same neuron to provide a range of adapting mechanisms. The presynaptic adaptation mechanism can also account for the contrast gain curves of cortical simple cells.

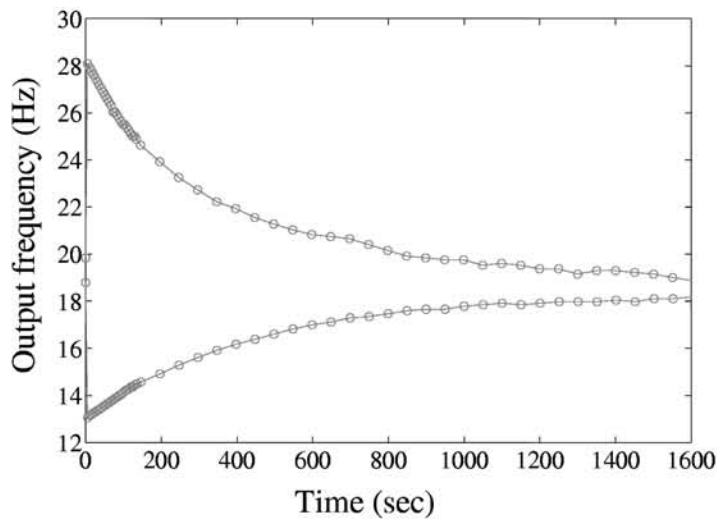

Figure 5: Response of silicon neuron to an increase and a decrease of a step input of 0.2V. The curve shows that the adaptation time constant is in the order of about 10 min.

**Acknowledgments**

We thank Rodney Douglas for supporting this work, the MOSIS foundation for fabricating this circuit, and Tobias Delbrück for proofreading this document. This work was supported in part by the Swiss National Foundation Research SPP grant and the U.S. Office of Naval Research.

## References

Desai, N., L. Rutherford, and G. Turrigiano (1999, Jun). Plasticity in the intrinsic excitability of cortical pyramidal neurons. *Nature Neuroscience 2*(6), 515–520.

Diorio, C., B. A. Minch, and P. Hasler (1999). Floating-gate MOS learning systems. *Proceedings of the International Symposium on the Future of Intellectual Integrated Electronics (ISFIIE)*, 515–524.

Lenzlinger, M. and E. H. Snow (1969). Fowler-Nordheim tunneling into thermally grown $SiO_2$. *Journal of Applied Physics 40*, 278–283.

Liu, Z., J. Golowasch, E. Marder, and L. Abbott (1998). A model neuron with activity-dependent conductances regulated by multiple calcium sensors. *Journal of Neuroscience 18*(7), 2309–2320.

Mead, C. (1989). *Analog VLSI and neural systems*. Reading, MA: Addison-Wesley.

Ohzawa, I., G. Sclar, and R. Freeman (1985). Contrast gain control in the cat's visual system. *Journal of Neurophys. 54*, 651–667.

Shin, J. and C. Koch (1999). Dynamic range and sensitivity adaptation in a silicon spiking neuron. *IEEE Trans. on Neural Networks 10*(5), 1232–1238.

Simoni, M. and S. DeWeerth (1999). Adaptation in an aVLSI model of a neuron. *IEEE CAS II-Analog and Digital Signal Processing 46*(7), 967–970.

Stemmler, M. and C. Koch (1999). How voltage-dependent conductances can adapt to maximize the information encoded by neuronal firing rate. *Nature Neuroscience 2*(6), 521–527.
